# On Sparsity and Overcompleteness in Image Models

**Pietro Berkes, Richard Turner, and Maneesh Sahani**
Gatsby Computational Neuroscience Unit, UCL
Alexandra House, 17 Queen Square, London WC1N 3AR

## Abstract

Computational models of visual cortex, and in particular those based on sparse coding, have enjoyed much recent attention. Despite this currency, the question of how sparse or how over-complete a sparse representation should be, has gone without principled answer. Here, we use Bayesian model-selection methods to address these questions for a sparse-coding model based on a Student-t prior. Having validated our methods on toy data, we find that natural images are indeed best modelled by extremely sparse distributions; although for the Student-t prior, the associated optimal basis size is only modestly over-complete.

## 1 Introduction

Computational models of visual cortex, and in particular those based on sparse coding, have recently enjoyed much attention. The basic assumption behind sparse coding is that natural scenes are composed of structural primitives (edges or lines, for example) and, although there are a potentially large number of these primitives, typically only a few are active in a single natural scene (hence the term sparse, [1, 2]). The claim is that cortical processing uses these statistical regularities to shape a representation of natural scenes, and in particular converts the pixel-based representation at the retina to a higher-level representation in terms of these structural primitives.

Traditionally, research has focused on determining the characteristics of the structural primitives and comparing their representational properties with those of V1. This has been a successful enterprise, but as a consequence other important questions have been neglected. The two we focus on here are: How large is the set of structural primitives best suited to describe all natural scenes (how over-complete), and how many primitives are active in a single scene (how sparse)? We will also be interested in the coupling between sparseness and over-completeness. The intuition is that, if there are a great number of structural primitives, they can be very specific and only a small number will be active in a visual scene. Conversely if there are a small number they have to be more general and a larger number will be active on average. We attempt to map this coupling by evaluating models with different over-completenesses and sparsenesses and discover where natural scenes live along this trade-off (see Fig. 1).

In order to test the sparse coding hypothesis it is necessary to build algorithms that both learn the primitives and decompose natural scenes in terms of them. There have been many ways to derive such algorithms, but one of the more successful is to regard the task of building a representation of natural scenes as one of probabilistic inference. More specifically, the unknown activities of the structural primitives are viewed as latent variables that must be inferred from the natural scene data. Commonly the inference is carried out by writing down a generative model (although see [3] for an alternative), which formalises the assumptions made about the data and latent variables. The rules of probability are then used to derive inference and learning algorithms.

Unfortunately the assumption that natural scenes are composed of a small number of structural primitives is not sufficient to build a meaningful generative model. Other assumptions must therefore be made and typically these are that the primitives occur independently, and combine linearly. These

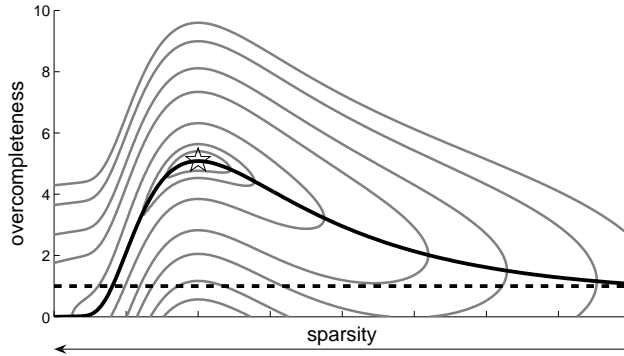

**Figure 1:** Schematic showing the space of possible sparse coding models in terms of sparseness (increasing in the direction of the arrow) and over-completeness. For reference, complete models lie along the dashed black line. Ideally every model could be evaluated (e.g. via their marginal likelihood or cross-validation) and the grey contours illustrate what we might expect to discover if this were possible: The solid black line illustrates the hypothesised trade-off between over-completeness and sparsity, whilst the star shows the optimal point in this trade-off.

are drastic approximations and it is an open question to what extent this affects the results of sparse coding. The distribution over the latent variables $x_{t,k}$ is chosen to be sparse and typical choices are Student-t, a Mixture of Gaussians (with zero means), and the Generalised Gaussian (which includes the Laplace distribution). The output $\mathbf{y}_t$ is then given by a linear combination of the $K$, $D$-dimensional structural primitives $\mathbf{g}_k$, weighted by their activities, plus some additive Gaussian noise (the model reduces to independent components analysis in the absence of this noise [4]),

$$p(x_{t,k}|\alpha) = p_{\text{sparse}}(\alpha) \tag{1}$$

$$p(\mathbf{y}_t|\mathbf{x}_t, \mathbf{G}) = \mathcal{N}_{\mathbf{y}_t}(\mathbf{G}\mathbf{x}_t, \boldsymbol{\Sigma}_y). \tag{2}$$

The goal of this paper will be to learn the optimal dimensionality of the latent variables ($K$) and the optimal sparseness of the prior ($\alpha$). In order to do this a notion of optimality has to be defined. One option is to train many different sparse-coding models and find the one which is most "similar" to visual processing. (Indeed this might be a fair characterisation of much of the current activity in field.) However, this is fraught with difficulty not least as it is unclear how recognition models map to neural processes. We believe the more consistent approach is, once again, to use the Bayesian framework and view this as a problem of probabilistic inference. In fact, if the hypothesis is that the visual system is implementing an optimal generative model, then questions of over-completeness and sparsity should be addressed in this context.

Unfortunately, this is not a simple task and quite sophisticated machine-learning algorithms have to be harnessed in order to answer these seemingly simple questions. In the first part of this paper we describe these algorithms and then validate them using artificial data. Finally, we present results concerning the optimal sparseness and over-completeness for natural image patches in the case that the prior is a Student-t distribution.

## 2   Model

As discussed earlier, there are many variants of sparse-coding. Here, we focus on the Student-t prior for the latent variables $x_{t,k}$:

$$p(x_{t,k}|\alpha, \lambda) = \frac{\Gamma\left(\frac{\alpha+1}{2}\right)}{\lambda\sqrt{\alpha\pi}\,\Gamma\left(\frac{\alpha}{2}\right)} \left(1 + \frac{1}{\alpha}\left(\frac{x_{t,k}}{\lambda}\right)^2\right)^{-\frac{\alpha+1}{2}} \tag{3}$$

There are two main reasons for this choice: The first is that this is a widely used model [1]. The second is that by implementing the Student-t prior using an auxiliary variable, all the distributions in the generative model become members of the exponential family [5]. This means it is easy to derive efficient approximate inference schemes like variational Bayes and Gibbs sampling.

The auxiliary variable method is based on the observation that a Student-t distribution is a continuous mixture of zero-mean Gaussians, whose mixing proportions are given by a Gamma distribution over

the precisions. This indicates that we can exchange the Student-t prior for a two-step prior in which we first draw a precision from a Gamma distribution and then draw an activation from a Gaussian with that precision,

$$p(u_{t,k}|\alpha,\lambda) = \mathcal{G}_{u_{t,k}}\left(\frac{\alpha}{2}, \frac{2}{\alpha\lambda^2}\right), \tag{4}$$

$$p(x_{t,k}|u_{t,k}) = \mathcal{N}_{x_{t,k}}\left(0, u_{t,k}^{-1}\right), \tag{5}$$

$$p(\mathbf{y}_t|\mathbf{x}_t, \mathbf{G}) = \mathcal{N}_{\mathbf{y}_t}(\mathbf{G}\mathbf{x}_t, \mathbf{\Sigma}_y), \tag{6}$$

$$\mathbf{\Sigma}_y := \mathrm{diag}(\sigma_y^2). \tag{7}$$

This model produces data which are often near zero, but occasionally highly non-zero. These non-zero elements form star-like patterns, where the points of the star are determined by the direction of the weights (e.g., Fig. 2).

One of the major technical difficulties posed by sparse-coding is that, in the over-complete regime, the posterior distribution of the latent variables $p(X|Y,\theta)$ is often complex and multi-modal. Approximation schemes are therefore required, but we must be careful to ensure that the scheme we choose does not bias the conclusions we are trying to draw. This is true for any application of sparse coding, but is particularly pertinent for our problem as we will be quantitatively comparing different sparse-coding models.

## 3 Bayesian Model Comparison

A possible strategy for investigating the sparseness/over-completeness coupling would be to tile the space with models and learn the parameters at each point (as schematised in Fig. 1). A model comparison criterion could then be used to rank the models, and to find the optimal sparseness/over-completeness. One such criterion would be to use cross validation and evaluate the likelihoods on some held-out test data. Another is to use (approximate) Bayesian Model Comparison, and it is on this method that we focus.

To evaluate the plausibility of two alternative versions of a model $\mathcal{M}$, each with a different setting of the hyperparameters $\Xi_1$ and $\Xi_2$, in the light of some data $Y$, we compute the evidence [6]:

$$\frac{p(\mathcal{M},\Xi_1|Y)}{p(\mathcal{M},\Xi_2|Y)} = \frac{p(Y|\mathcal{M},\Xi_1)\,P(\mathcal{M},\Xi_1)}{p(Y|\mathcal{M},\Xi_2)\,P(\mathcal{M},\Xi_2)}. \tag{8}$$

Since we do not have any reason *a priori* to prefer one particular configuration of hyperparameters to another, we take the prior terms $P(\mathcal{M},\Xi_i)$ to be equal, which leaves us with the ratio of the marginal-likelihoods (or Bayes Factor),

$$\frac{P(Y|\mathcal{M},\Xi_1)}{P(Y|\mathcal{M},\Xi_2)}, \tag{9}$$

The marginal-likelihoods themselves are hard to compute, being formed from high dimensional integrals over the latent variables $V$ and parameters $\Theta$,

$$p(Y|\mathcal{M},\Xi_i) = \int \mathrm{d}V\mathrm{d}\Theta\; p(Y,V,\Theta|\mathcal{M},\Xi_i) \tag{10}$$

$$= \int \mathrm{d}V\mathrm{d}\Theta\; p(Y,V|\Theta,\mathcal{M},\Xi_i)p(\Theta|\mathcal{M},\Xi_i). \tag{11}$$

One concern in model comparison might be that the more complex models (those which are more over-complete) have a larger number parameters and therefore 'fit' any data set better. However, the Bayes factor (Eq. 9) implicitly implements a probabilistic version of Occam's razor that penalises more complex models and mitigates this effect [6]. This makes the Bayesian method appealing for determining the over-completeness of a sparse-coding model.

Unfortunately computing the marginal-likelihood is computationally intensive, and this precludes tiling the sparseness/over-completeness space. However, an alternative is to learn the optimal over-completeness at a given sparseness using automatic relevance determination (ARD) [7, 8]. The

advantage of ARD is that it changes a hard and lengthy model comparison problem (i.e., computing the marginal-likelihood for many models of differing dimensionalities) into a much simpler inference problem. In a nutshell, the idea is to equip the model with many more components than are believed to be present in the data, and to let it prune out the weights which are unnecessary. Practically this involves placing a (Gaussian) prior over the components which favours small weights, and then inferring the scale of this prior. In this way the scale of the superfluous weights is driven to zero, removing them from the model. The necessary ARD hyper-priors are

$$p(\mathbf{g}_k|\gamma_k) = \mathcal{N}_{\mathbf{g}_k}(\mathbf{0}, \gamma_k^{-1}) \,, \tag{12}$$

$$p(\gamma_k) = \mathcal{G}_{\gamma_k}(\theta_k, l_k) \,. \tag{13}$$

## 4  Determining the over-completeness: Variational Bayes

In the previous two sections we described a generative model for sparse coding that is theoretically able to learn the optimal over-completeness of natural scenes. We have two distinct uses for this model: The first, and computationally more demanding task, is to learn the over-completeness at a variety of different, fixed, sparsenesses (that is, to find the optimal over-completeness in a vertical slice through Fig. 1); The second is to determine the optimal point on this trade-off by evaluating the (approximate) marginal-likelihood (that is, evaluating points along the trade-off line in Fig. 1 to find the optimal model - the star). It turns out that no single method is able to solve both these tasks, but that it is possible to develop a pair of approximate algorithms to solve them separately. The first approximation scheme is Variational Bayes (VB), and it excels at the first task, but is severely biased in the case of the second. The second scheme is Annealed Importance Sampling (AIS) which is prohibitively slow for the first task, but much more accurate on the second. We describe them in turn, starting with VB.

The quantity required for learning is the marginal-likelihood,

$$\log p(Y|\mathcal{M}, \Xi) = \log \int dV d\Theta \, p(Y, V, \Theta|\mathcal{M}, \Xi). \tag{14}$$

Computing this integral is intractable (for reasons similar to those given in Sec. 2), but a lower-bound can be constructed by introducing *any* distribution over the latent variables and parameters, $q(V, \Theta)$, and using Jensen's inequality,

$$\log p(Y|\mathcal{M}, \Xi) \geq \int dV d\Theta \, q(V, \Theta) \log \frac{p(Y, V, \Theta|\mathcal{M}, \Xi)}{q(V, \Theta)} =: \mathcal{F}(q(V, \Theta)) \tag{15}$$

$$= \log p(Y|\mathcal{M}, \Xi) - KL(q(V, \Theta)||p(V, \Theta|Y)) \tag{16}$$

This lower-bound is called the free-energy, and the idea is to repeatedly optimise it with respect to the distribution $q(V, \Theta)$ so that it becomes as close to the true marginal likelihood as possible. Clearly the optimal choice for $q(V, \Theta)$ is the (intractable) true posterior. However, by constraining this distribution headway can be made. In particular if we assume that the set of parameters and set of latent variables are independent in the posterior, so that $q(V, \Theta) = q(V)q(\Theta)$ then we can sequentially optimise the free-energy with respect to each of these distributions. For large hierarchical models, including the one described in this paper, it is often necessary to introduce further factorisations within these two distributions in order to derive the updates. Their general form is,

$$q(V_i) \propto \exp \langle \log p(V, \Theta) \rangle_{q(\Theta) \prod_{j \neq i} q(V_i)} \tag{17}$$

$$q(\Theta_i) \propto \exp \langle \log p(V, \Theta) \rangle_{q(V) \prod_{j \neq i} q(\Theta_i)} \,. \tag{18}$$

As the Bayesian Sparse Coding model is composed of distributions from the exponential family, the functional form of these updates is the same as the corresponding priors. So, for example the latent variables have the following form: $q(\mathbf{x}_t)$ is Gaussian and $q(u_{t,k})$ is Gamma distributed.

Although this approximation is good at discovering the over-completeness of data at fixed sparsities, it provides an estimate of the marginal-likelihood (the free-energy) which is biased toward regions of low sparsity. The reason is simple to understand. The difference between the free energy and the true likelihood is given by the KL divergence between the approximate and true posterior. Thus, the free-energy bound is tightest in regions where $q(V, \Theta)$ is a good match to the true posterior, and loosest in

regions where it is a poor match. At high sparsities, the true posterior is multimodal and highly non-Gaussian. In this regime $q(V, \Theta)$ – which is always uni-modal – is a poor approximation. At low-sparsities the prior becomes Gaussian-like and the posterior also becomes a uni-modal Gaussian. In this regime $q(V, \Theta)$ is an excellent approximation. This leads to a consistent bias in the peak of the free-energy toward regions of low sparsity. One might also be concerned with another potential source of bias: The number of modes in the posterior increases with the number of components in the model, which gives a worse match to the variational approximation for more over-complete models. However, because of the sparseness of the prior distribution, most of the modes are going to be very shallow for typical inputs, so that this effect should be small. We verify this claim on artificial data in Section 6.2.

## 5 Determining the sparsity: AIS

An approximation scheme is required to estimate the marginal-likelihood, but without a sparsity-dependent bias. Any scheme which uses a uni-modal approximation to the posterior will inevitably fall victim to such biases. This rules out many alternate variational schemes, as well as methods like the Laplace approximation, or Expectation Propagation. One alternative might be to use a variational method which has a multi-modal approximating distribution (e.g. a mixture model). The approach taken here is to use Annealed Importance Sampling (AIS) [9] which is one of the few methods for evaluating normalising constants of intractable distributions. The basic idea behind AIS is to estimate the marginal-likelihood using importance sampling. The twist is that the proposal distribution for the importance sampler is itself generated using an MCMC method. Briefly, this inner loop starts by drawing samples from the model's prior distribution and continues to sample as the prior is deformed into the posterior, according to an annealing schedule. Both the details of this schedule, and having a quick-mixing MCMC method, are critical for good results. In fact it is simple to derive a quick-mixing Gibbs sampler for our application and this makes AIS particularly appealing.

## 6 Results

Before tackling natural images, it is necessary to verify that the approximations can discover the correct degree of over-completeness and sparsity in the case where the data are drawn from the forward model. This is done in two stages: Firstly we focus on a very simple, low-dimensional example that is easy to visualise and which helps explicate the learning algorithms, allowing them to be tuned; Secondly, we turn to a larger scale example designed to be as similar to the tests on natural data as possible.

### 6.1 Verification using simple artifical data

In the first experiment the training data are produced as follows: Two-dimensional observations are generated by three Student-t sources with degree of freedom chosen to be $2.5$. The generative weights are fixed to be 60 degrees apart from one another, as shown in Figure 2.

A series of VB simulations were then run, differing only in the sparseness level (as measured by the degrees of freedom of the Student-t distribution over $\mathbf{x}_t$). Each simulation consisted of 500 VB iterations performed on a set of 3000 data points randomly generated from the model. We initialised the simulations with $K = 7$ components. To improve convergence, we started the simulations with weights near the origin (drawn from a normal distribution with mean 0 and standard deviation $10^{-8}$) and a relatively large input noise variance, and annealed the noise variance between the iterations of VBEM. The annealing schedule was as following: we started with $\sigma_y^2 = 0.3$ for 100 iterations, reduced this linearly down to $\sigma_y^2 = 0.1$ in 100 iterations, and finally to $\sigma_y^2 = 0.01$ in a further 50 iterations. During the annealing process, the weights typically grew from the origin and spread in all directions to cover the input space. After an initial growth period, where the representation usually became as over-complete as allowed by the model, some of the weights rapidly shrank again and collapsed to the origin. At the same time, the corresponding precision hyperparameters grew and effectively pruned the unnecessary components. We performed 7 blocks of simulations at different sparseness levels. In every block we performed 3 runs of the algorithm and retained the result with the highest free energy.

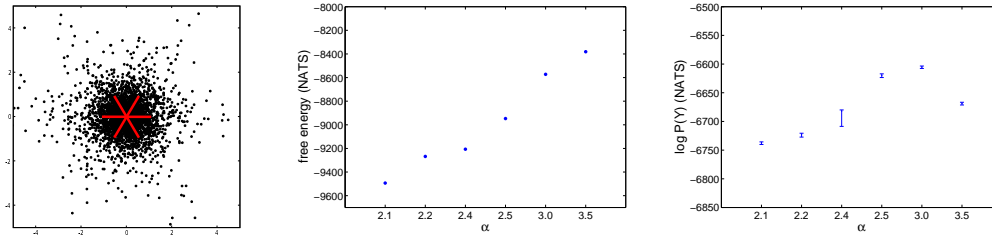

**Figure 2:** Left: Test data drawn from the simple artificial model. Centre: Free energy of the models learned by VBEM in the artificial data case. Right: Estimated log marginal likelihood. Error bars are 3 times the estimated standard deviation.

The marginal likelihoods of the selected results were then estimated using AIS. We derived the importance weights using a fixed data set with 2500 data points, 250 samples, and 300 intermediate distributions. Following the recommendations in [9], the annealing schedule was chosen to be linear initially (with 50 inverse temperatures spaced uniformly from 0 to 0.01), followed by a geometric section (250 inverse temperatures spaced geometrically from 0.01 to 1). This mean that there were a total of 300 distributions between the prior and posterior.

The results indicate that the combination of the two methods is successful at learning both the over-completeness and sparseness. In particular the VBEM algorithm was able to recover the correct dimensionality for all sparseness levels, except for the sparsest case $\alpha = 2.1$, where it preferred a model with 5 significant components. As expected, however, figure 2 shows that the maximum free energy is biased toward the more Gaussian models. In contrast to this, the marginal likelihood estimated by AIS (Fig. 2), which is strictly greater than the free-energy as expected, favours sparseness levels close to the true value.

## 6.2  Verification using complex artificial data

Although it is necessary that the inference scheme should pass simple tests like that in the previous section, they are not sufficient to give us confidence that it will perform successfully on natural data. One pertinent criticism is that the regime in which we tested the algorithms in the previous section (two dimensional observations, and three hidden latents) is quite different from that required to model natural data. To that end, in this section we first learn a sparse model for natural images with fixed over-completeness levels using a Maximum A Posteriori (MAP) algorithm [2] (degree of freedom 2.5). These solutions are then used to generate artificial data as in the previous section. The goal is to validate the model on data which has a content and scale similar to the natural images case, but with a controlled number of generative components.

The image data comprised patches of size $9 \times 9$ pixels, taken at random positions from 36 natural images randomly selected from the van Hateren database (preprocessed as described in [10]). The patches were whitened and their dimensionality reduced from 81 to 36 by principal component analysis. The MAP solution was trained for 500 iterations, with every iteration performed on a new batch of 1440 patches (100 patches per image).

The model was initialised with a 3-times over-complete number of components ($K = 108$). As above, the weights were initialised near the origin, and the input noise was annealed linearly from $\sigma_d = 0.5$ to $\sigma_d = 0.2$ in the first 300 iterations, remaining constant thereafter. Every run consisted of 500 VBEM iterations, with every iteration performed on 3600 patches generated from the MAP solution. We performed several simulations for over-completeness levels between 0.5 and 4.5, and retained the solutions with the highest free energy.

The results are summarised in Figure 3: The model is able to recover the underlying dimensionality for data between 0.5 and 2 times over-complete, and correctly saturates to 3 times over-complete (the maximum attainable level here) when the data over-completeness exceeds 3. In the regime between 2.5 and 3 times over-complete data, the model returns solutions with a smaller number of components, which is possibly due to the bias described at the end of Section 5. However, these

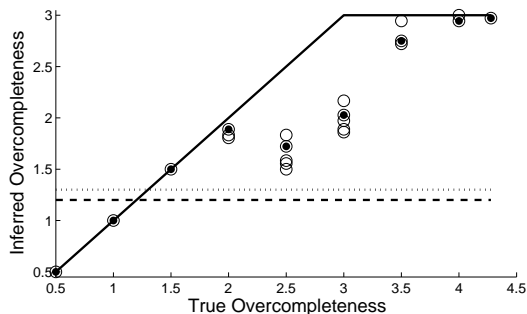

**Figure 3:** True versus inferred over-completeness from data drawn from the forward model trained on natural images. If inference was perfect, the true over-completeness would be recovered (black line). This straight line saturates when we hit the number of latent variables with which ARD was initialised (three times over-complete). The results using multiple runs of ARD are close to this line (open circles, simulations with the highest free-energy are shown as closed circles). The maximal and best over-completeness inferred from natural scenes is shown by the dotted line, and lies well below the over-completenesses we are able to infer.

values are still far above the highest over-completeness learned from natural images (see section 6.3), so that we believe that the bias does not invalidate our conclusions.

## 6.3 Natural images

Having established that the model performs as expected, at least when the data is drawn from the forward model, we now turn to natural image data and examine the optimal over-completeness ratio and sparseness degree for natural scene statistics.

The image data for this simulation and the model initialisation and annealing procedure are identical to the ones in the experiments in the preceeding section. We performed 20 simulations with different sparseness levels, especially concentrated on the more sparse values. Every run comprised 500 VBEM iterations, with every iteration performed on a new batch of 3600 patches.

As shown in Figure 4, the free energy increased almost monotonically until $\alpha = 5$ and then stabilised and started to decrease for more Gaussian models. The algorithm learnt models that were only slightly over-complete: the over-completeness ratio was distributed between 1 and 1.3, with a trend for being more over-complete at high sparseness levels (Fig. 4). Although this general trend accords with the intuition that sparseness and over-completeness are coupled, both the magnitude of the effect and the degree of over-completeness is smaller than might have been anticipated. Indeed, this result suggests that highly over-complete models with a Student-t prior may very well be overfitting the data.

Finally we performed AIS using the same annealing schedule as in Section 6.1, using 250 samples for the first 6 sparseness levels and 50 for the successive 14. The estimates obtained for the log marginal likelihood, shown in Figure 4, were monotonically *increasing* with increasing sparseness (decreasing $\alpha$). This indicates that sparse models are indeed optimal for natural scenes. Note that this is exactly the opposite trend to that of the free energy, indicating that it is also biased for natural scenes. Figure 4 shows the basis vectors learned in the simulation with $\alpha = 2.09$, which had maximal marginal likelihood. The weights resemble the Gabor wavelets, typical of sparse codes for natural images [1].

## 7 Discussion

Our results suggest that the optimal sparse-coding model for natural scenes is indeed one which is very sparse, but only modestly over-complete. The anticipated coupling between the degree of sparsity and the over-completeness in the model is visible, but is weak.

One crucial question is how far these results will generalise to other prior distributions; and indeed, which of the various possible sparse-coding priors is best able to capture the structure of natural scenes. One indication that the Student-t might not be optimal, is its behaviour as the degree-of-

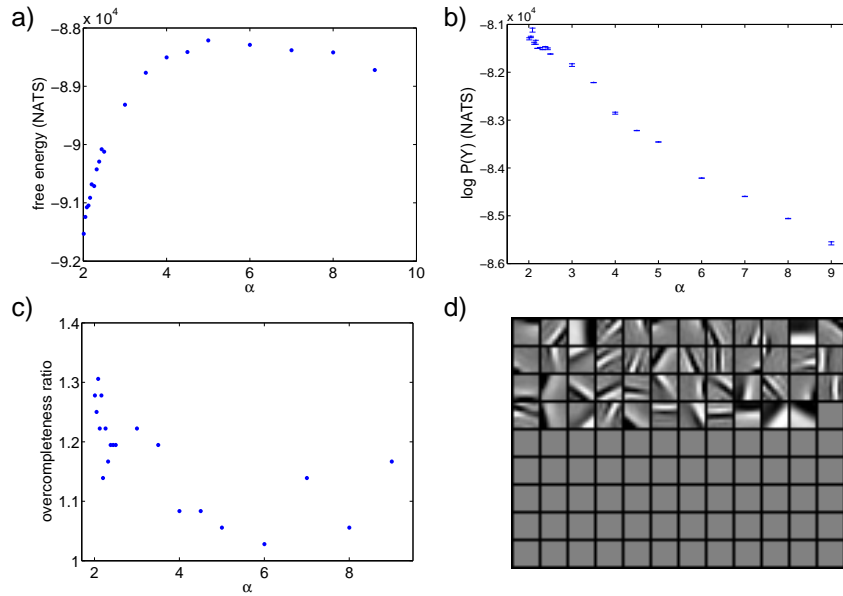

**Figure 4:** Natural images results. a) Free energy b) Marginal likelihood c) Estimated over-completeness d) Basis vectors

freedom parameter moves towards sparser values. The distribution puts a very small amount of mass at a very great distance from the mean (for example, the kurtosis is undefined for $\alpha < 4$). It is not clear that data with such extreme values will be encountered in typical data sets, and so the model may become distorted at high sparseness values.

Future work will be directed towards more general prior distributions. The formulation of the Student-t in terms of a random precision Gaussian is computationally helpful. While no longer within the exponential family, other distributions on the precision (such as a uniform one) may be approximated using a similar approach.

### Acknowledgements

This work has been supported by the Gatsby Charitable Foundation. We thank Yee Whye Teh, Iain Murray, and David McKay for fruitful discussions.

### References

[1] B.A. Olshausen and D.J. Field. Emergence of simple-cell receptive field properties by learning a sparse code for natural images. *Nature*, 381(6583):607–609, 1996.

[2] B.A. Olshausen and D.J. Field. Sparse coding with an overcomplete basis set: A strategy employed by V1? *Vision Research*, 37:3311–3325, 1997.

[3] Y.W Teh, M. Welling, S. Osindero, and G.E. Hinton. Energy-based models for sparse overcomplete representations. *Journal of Machine Learning Research*, 4:1235–1260, 2003.

[4] A.J. Bell and T.J. Sejnowski. The 'independent components' of natural scenes are edge filters. *Vision Research*, 37(23):3327–3338, 1997.

[5] S. Osindero, M. Welling, and G.E. Hinton. Topographic product models applied to natural scene statistics. *Neural Computation*, 18:381–344, 2006.

[6] D.J.C. McKay. Bayesian interpolation. *Neural Comput*, 4(3):415–447, 1992.

[7] C.M. Bishop. Variational principal components. In *ICANN 1999 Proceedings*, pages 509–514, 1999.

[8] M.J. Beal. *Variational Algorithms for Approximate Bayesian Inference*. PhD thesis, Gatsby Computational Neuroscience Unit, University College London, 2003.

[9] R.M. Neal. Annealed importance sampling. *Statistics and Computing*, 11:125–139, 2001.

[10] J.H. van Hateren and A. van der Schaaf. Independent component filters of natural images compared with simple cells in primary visual cortex. *Proc. R. Soc. Lond. B*, 265:359–366, 1998.

